# Bayesian Methods for Mixtures of Experts

**Steve Waterhouse**
Cambridge University
Engineering Department
Cambridge CB2 1PZ
England
Tel: [+44] 1223 332754
srw1001@eng.cam.ac.uk

**David MacKay**
Cavendish Laboratory
Madingley Rd.
Cambridge CB3 0HE
England
Tel: [+44] 1223 337238
mackay@mrao.cam.ac.uk

**Tony Robinson**
Cambridge University
Engineering Department
Cambridge CB2 1PZ
England.
Tel: [+44] 1223 332815
ajr@eng.cam.ac.uk

## ABSTRACT

We present a Bayesian framework for inferring the parameters of a mixture of experts model based on ensemble learning by variational free energy minimisation. The Bayesian approach avoids the over-fitting and noise level under-estimation problems of traditional maximum likelihood inference. We demonstrate these methods on artificial problems and sunspot time series prediction.

## INTRODUCTION

The task of estimating the parameters of adaptive models such as artificial neural networks using Maximum Likelihood (ML) is well documented eg. Geman, Bienenstock & Doursat (1992). ML estimates typically lead to models with high variance, a process known as "over-fitting". ML also yields over-confident predictions; in regression problems for example, ML underestimates the noise level. This problem is particularly dominant in models where the ratio of the number of data points in the training set to the number of parameters in the model is low. In this paper we consider inference of the parameters of the hierarchical mixture of experts (HME) architecture (Jordan & Jacobs 1994). This model consists of a series of "experts," each modelling different processes assumed to be underlying causes of the data. Since each expert may focus on a different subset of the data which may be arbitrarily small, the possibility of over-fitting of each process is increased. We use Bayesian methods (MacKay 1992a) to avoid over-fitting by specifying prior belief in various aspects of the model and marginalising over parameter uncertainty.

The use of regularisation or "weight decay" corresponds to the prior assumption that the model should have smooth outputs. This is equivalent to a prior $P(\theta|\alpha)$ on the parameters $\theta$ of the model, where $\alpha$ are the *hyperparameters* of the prior. Given a set of priors we may specify a *posterior* distribution of the parameters given data $D$,

$$P(\theta|D, \alpha, \mathcal{H}) \propto P(D|\theta, \mathcal{H})P(\theta|\alpha, \mathcal{H}), \tag{1}$$

where the variable $\mathcal{H}$ encompasses the assumptions of model architecture, type of regularisation used and assumed noise model. Maximising the posterior gives us the *most probable* parameters $\theta_{MP}$. We may then set the hyperparameters either by cross-validation, or by finding the maximum of the posterior distribution of the

hyperparameters $P(\alpha|D)$, also known as the "evidence" (Gull 1989). In this paper we describe a method, motivated by the Expectation Maximisation (EM) algorithm of Dempster, Laird & Rubin (1977) and the principle of ensemble learning by variational free energy minimisation (Hinton & van Camp 1993, Neal & Hinton 1993) which achieves simultaneous optimisation of the parameters *and* hyperparameters of the HME. We then demonstrate this algorithm on two simulated examples and a time series prediction task. In each task the use of the Bayesian methods prevents over-fitting of the data and gives better prediction performance. Before we describe this algorithm, we will specify the model and its associated priors.

## MIXTURES OF EXPERTS

The mixture of experts architecture (Jordan & Jacobs 1994) consists of a set of "experts" which perform local function approximation. The expert outputs are combined by a "gate" to form the overall output. In the hierarchical case, the experts are themselves mixtures of further experts, thus extending the network in a tree structured fashion. The model is a generative one in which we assume that data are generated in the domain by a series of $J$ independent processes which are selected in a stochastic manner. We specify a set of *indicator* variables $Z = \{z_j^{(n)} : j = 1 \ldots J, n = 1 \ldots N\}$, where $z_j^{(n)}$ is 1 if the output $y^{(n)}$ was generated by expert $j$ and zero otherwise. Consider the case of regression over a data set $D = \{\boldsymbol{x}^{(n)} \in \mathfrak{R}^k, y^{(n)} \in \mathfrak{R}^p, n = 1 \ldots N\}$ with $p = 1$. We specify that the conditional probability of the scalar output $y^{(n)}$ given the input vector $\boldsymbol{x}^{(n)}$ at exemplar $(n)$ is

$$P(y^{(n)}|\boldsymbol{x}^{(n)}, \theta) = \sum_{j=1}^{J} P(z_j^{(n)}|\boldsymbol{x}^{(n)}, \xi_j)P(y^{(n)}|\boldsymbol{x}^{(n)}, \mathbf{w}_j, \beta_j), \quad (2)$$

where $\{\xi_j \in \mathfrak{R}^k\}$ is the set of gate parameters, and $\{(\mathbf{w}_j \in \mathfrak{R}^k), \beta_j\}$ the set of expert parameters. In this case, $P(y^{(n)}|\boldsymbol{x}^{(n)}, \mathbf{w}_j, \beta_j)$ is a Gaussian:

$$\phi_j^{(n)} \equiv P(y^{(n)}|\boldsymbol{x}^{(n)}, \mathbf{w}_j, \beta_j) = \left(\frac{2\pi}{\beta_j}\right)^{-\frac{1}{2}} \exp\left(-\frac{\beta_j}{2}\left(y^{(n)} - \hat{y}_j^{(n)}\right)^2\right), \quad (3)$$

where $1/\beta_j$ is the variance of expert $j$,[1] and $\hat{y}_j^{(n)} = f_j(\boldsymbol{x}^{(n)}, \mathbf{w}_j)$ is the output of expert $j$, giving a probabilistic mixture model. In this paper we restrict the expert output to be a linear function of the input, $f_j(\boldsymbol{x}^{(n)}, \mathbf{w}_j) = \mathbf{w}_j^T \boldsymbol{x}^{(n)}$. We model the action of selecting process $j$ with the gate, the outputs of which are given by the softmax function of the inner products of the input vector[2] and the gate parameter vectors. The conditional probability of selecting expert $j$ given input $\boldsymbol{x}^{(n)}$ is thus:

$$g_j^{(n)} \equiv P(z_j^{(n)} = 1|\boldsymbol{x}^{(n)}, \xi_j) = \exp(\xi_j^T \boldsymbol{x}^{(n)}) \Bigg/ \sum_{i=1}^{J} \exp(\xi_i^T \boldsymbol{x}^{(n)}) \quad (4)$$

A straightforward extension of this model also gives us the conditional probability $h_j^{(n)}$ of expert $j$ having been selected given input $\boldsymbol{x}^{(n)}$ *and* output $y^{(n)}$,

$$h_j^{(n)} \equiv P(z_j^{(n)} = 1|y^{(n)}, \boldsymbol{x}^{(n)}, \theta) = g_j^{(n)}\phi_j^{(n)} \Bigg/ \sum_{i=1}^{J} g_i^{(n)}\phi_i^{(n)}. \quad (5)$$

## PRIORS

We assume a separable prior on the parameters $\theta$ of the model:

$$P(\theta|\alpha) = \prod_{j=1}^{J} P(\xi_j|\mu)P(\mathbf{w}_j|\alpha_j) \tag{6}$$

where $\{\alpha_j\}$ and $\{\mu\}$ are the hyperparameters for the parameter vectors of the experts and the gate respectively. We assume Gaussian priors on the parameters of the experts $\{\mathbf{w}_j\}$ and the gate $\{\xi_j\}$, for example:

$$P(\mathbf{w}_j|\alpha_j) = \left(\frac{\alpha_j}{2\pi}\right)^{\frac{k}{2}} \exp\left(-\frac{\alpha_j}{2}\mathbf{w}_j^T\mathbf{w}_j\right) \tag{7}$$

For simplicity of notation, we shall refer to the set of all smoothness hyperparameters as $\alpha = \{\mu, \alpha_j\}$ and the set of all noise level hyperparameters as $\beta = \{\beta_j\}$.

Finally, we assume Gamma priors on the hyperparameters $\{\mu, \alpha_j, \beta_j\}$ of the priors, for example:

$$P(\log \beta_j|\rho_\beta, \upsilon_\beta) = \frac{1}{\Gamma(\rho_\beta)} \left(\frac{\beta_j}{\upsilon_\beta}\right)^{\rho_\beta} \exp(-\beta_j / \upsilon_\beta), \tag{8}$$

where $\upsilon_\beta, \rho_\beta$ are the *hyper*-hyperparameters which specify the range in which we expect the noise levels $\beta_j$ to lie.

## INFERRING PARAMETERS USING ENSEMBLE LEARNING

The EM algorithm was used by Jordan & Jacobs (1994) to train the HME in a maximum likelihood framework. In the EM algorithm we specify a complete data set $\{D, Z\}$ which includes the observed data $D$ and the set of indicator variables $Z$. Given $\theta^{(m-1)}$, the E step of the EM algorithm computes a distribution $P(Z|D, \theta^{(m-1)})$ over $Z$. The M step then maximises the expected value of the *complete* data likelihood $P(D, Z|\theta)$ over this distribution. In the case of the HME, the indicator variables $Z = \{\{z_j^{(n)}\}_{j=1}^J\}_{n=1}^N$ specify which expert was responsible for generating the data at each time.

We now outline an algorithm for the simultaneous optimisation of the parameters $\theta$ and hyperparameters $\alpha$ and $\beta$, using the framework of ensemble learning by variational free energy minimisation (Hinton & van Camp 1993). Rather than optimising a point estimate of $\theta$, $\alpha$ and $\beta$, we optimise a distribution over these parameters. This builds on Neal & Hinton's (1993) description of the EM algorithm in terms of variational free energy minimisation.

We first specify an approximating *ensemble* $Q(\mathbf{w}, \xi, \alpha, \beta, Z)$ which we optimise so that it approximates the posterior distribution $P(\mathbf{w}, \xi, \alpha, \beta, Z|D, \mathcal{H})$ well. The objective function chosen to measure the quality of the approximation is the *variational free energy*,

$$F(Q) = \int d\mathbf{w} \, d\xi \, d\alpha \, d\beta \, dZ \, Q(\mathbf{w}, \xi, \alpha, \beta, Z) \log \frac{Q(\mathbf{w}, \xi, \alpha, \beta, Z)}{P(\mathbf{w}, \xi, \alpha, \beta, Z, D|\mathcal{H})}, \tag{9}$$

where the joint probability of parameters $\{\mathbf{w}, \xi\}$, hyperparameters, $\{\alpha, \beta\}$, missing data $Z$ and observed data $D$ is given by,

$$P(\mathbf{w}, \xi, \alpha, \beta, Z, D|\mathcal{H}) =$$

$$P(\mu)\prod_{j=1}^{J} P(\xi_j|\mu)P(\alpha_j)P(\mathbf{w}_j|\alpha_j)P(\beta_j|\rho_j, \upsilon_j)\prod_{n=1}^{N}\left(P(z_j^{(n)}=1|\boldsymbol{x}^{(n)}, \xi_j)P(y^{(n)}|\boldsymbol{x}^{(n)}, \mathbf{w}_j, \beta_j)\right)^{z_j^{(n)}} \quad (10)$$

The free energy can be viewed as the sum of the negative log evidence $-\log P(D|\mathcal{H})$ and the Kullback-Leibler divergence between $Q$ and $P(\mathbf{w}, \xi, \alpha, \beta, Z|D, \mathcal{H})$. $F$ is bounded below by $-\log P(D|\mathcal{H})$, with equality when $Q = P(\mathbf{w}, \xi, \alpha, \beta, Z|D, \mathcal{H})$.

We constrain the approximating ensemble $Q$ to be separable in the form $Q(\mathbf{w}, \xi, \alpha, \beta, Z) = Q(\mathbf{w})Q(\xi)Q(\alpha)Q(\beta)Q(Z)$. We find the optimal separable distribution $Q$ by considering separately the optimisation of $F$ over each separate ensemble component $Q(\cdot)$ with all other components fixed.

**Optimising $Q_w(\mathbf{w})$ and $Q_\xi(\xi)$.**

As a functional of $Q_w(\mathbf{w})$, $F$ is

$$F = \int d\mathbf{w}\, Q_w(\mathbf{w})\left[\sum_j \frac{\bar{\alpha}_j}{2}\mathbf{w}_j^T\mathbf{w}_j + \sum_{n=1}^{N}\bar{z}_j^{(n)}\frac{\bar{\beta}_j}{2}(y^{(n)} - \hat{y}_j^{(n)})^2 + \log Q_w(\mathbf{w})\right] + \text{const} \quad (11)$$

where for any variable $a$, $\bar{a}$ denotes $\int da\, Q(a)\, a$ . Noting that the $\mathbf{w}$ dependent terms are the log of a posterior distribution and that a divergence $\int Q\log(Q/\tilde{P})$ is minimised by setting $Q = \tilde{P}$, we can write down the distribution $Q_w(\mathbf{w})$ that minimises this expression. For given data and $Q_\alpha, Q_\beta, Q_z, Q_\xi$, the optimising distribution $Q_w^{\mathrm{opt}}(\mathbf{w})$ is

$$Q_w^{\mathrm{opt}}(\mathbf{w}) = \prod_j Q_{w_j}^{\mathrm{opt}}(w_j) = \prod_j \exp\left(-\frac{\bar{\alpha}_j}{2}\mathbf{w}_j^T\mathbf{w}_j - \sum_n \bar{z}_j^{(n)}\frac{\bar{\beta}_j}{2}(y^{(n)} - \hat{y}_j^{(n)})^2\right)\bigg/ \text{const} \quad (12)$$

This is a set of $J$ Gaussian distributions with means $\{\bar{w}_j\}$, which can be found exactly by quadratic optimisation. We denote the variance covariance matrices of $Q_{w_j}^{\mathrm{opt}}(w_j)$ by $\{\Sigma_{w_j}\}$. The analogous expression for the gates $Q_\xi^{\mathrm{opt}}(\xi)$ is obtained in a similar fashion and is given by

$$Q_\xi^{\mathrm{opt}}(\xi) = \prod_j Q_{\xi_j}^{\mathrm{opt}}(\xi_j) = \prod_j \exp\left(-\frac{\bar{\mu}_j}{2}\xi_j^T\xi_j + \sum_n \bar{z}_j^{(n)}\log g_j^{(n)}\right)\bigg/ \text{const.} \quad (13)$$

We approximate each $Q_{\xi_j}^{\mathrm{opt}}(\xi_j)$ by a Gaussian distribution fitted at its maximum $\xi_j = \bar{\xi}_j$ with variance covariance matrix $\Sigma_{\xi_j}$.

**Optimising $Q_z(Z)$**

By a similar procedure, the optimal distribution $Q_z^{\mathrm{opt}}(Z)$ is given by

$$Q_z^{\mathrm{opt}}(Z) = \prod_n \prod_j \left\{\exp(s_j^{(n)})\bigg/\sum_{i=1}^{J}\exp(s_i^{(n)})\right\} \quad (14)$$

$$\text{where}\quad s_j^{(n)} = \bar{\xi}_j^T \boldsymbol{x}^{(n)} - \frac{\bar{\beta}_j}{2}\left[(y^{(n)} - \hat{y}_j^{(n)}(\mathbf{w}_j^{MP}))^2 + \boldsymbol{x}^{(n)}\Sigma_{w_j}^{-1}\boldsymbol{x}^{(n)}\right] \quad (15)$$

and $\bar{\xi}_j$ is the value of $\xi_j$ computed above. The standard E-step gives us a distribution of $Z$ given a fixed value of parameters and the data, as shown in equation (5). In this case, by finding the optimal $Q_z(Z)$ we obtain the alternative expression of (15), with dependencies on the uncertainty of the experts' predictions. Ideally (if we did not made the assumption of a separable distribution $Q$) $Q_z$ might be expected to contain an additional effect of the uncertainty in the gate parameters. We can introduce this by the method of MacKay (1992b) for marginalising classifiers, in the case of binary gates.

**Optimising $Q_\alpha(\alpha)$ and $Q_\beta(\beta)$**

Finally, for the hyperparameter distributions, the optimal values of ensemble functions give values for $\alpha_j$ and $\beta_j$ as

$$\frac{1}{\bar{\alpha}_j} = \frac{\mathbf{w}_j^T \mathbf{w}_j + 2 / v_{\alpha_j} + \mathrm{Trace}\Sigma_{\mathbf{w}_j}}{k + 2\rho_{\alpha_j}}, \qquad \frac{1}{\bar{\beta}_j} = \frac{\sum_n \left[ \bar{z}_j^{(n)}(y^{(n)} - \hat{y}_j^{(N+1)})^2 + x^{(n)^T}\Sigma_{\mathbf{w}_j} x^{(n)} \right] + 2 / v_{\beta_j}}{\sum_n \bar{z}_j^{(n)} + 2\rho_{\beta_j}}. \quad (16)$$

An analogous procedure is used to set the hyperparameters $\{\mu\}$ of the gate.

## MAKING PREDICTIONS

In order to make predictions using the model, we must *marginalise* over the parameters and hyperparameters to get the predictive distribution. We use the optimal distributions $Q^{\mathrm{opt}}(\cdot)$ to approximate the posterior distribution.

For the experts, the marginalised outputs are given by $\bar{y}_j^{(N+1)} = f_j(x^{(N+1)}, \mathbf{w}_j^{MP})$, with variance $\sigma_{y|\alpha_j,\beta_j}^2 = x^{(N+1)^T}\Sigma_{w_j} x^{(N+1)} + \sigma_j^2$, where $\sigma_j^2 = 1 / \bar{\beta}_j$. We may also marginalise over the gate parameters (MacKay 1992b) to give marginalised outputs for the gates. The predictive distribution is then a mixture of Gaussians, with mean and variance given by its first and second moments,

$$\bar{y}^{(N+1)} = \sum_{i=1}^{J} g_i^{(N+1)} \bar{y}_i^{(N+1)}; \qquad \sigma_{y|\alpha,\beta}^2 = \sum_{i=1}^{J} g_i^{(N+1)}(\sigma_{y|\alpha_i,\beta_i}^2 + (\bar{y}_i^{(N+1)})^2) - (\bar{y}^{(N+1)})^2. \quad (17)$$

## SIMULATIONS
### Artificial Data

In order to test the performance of the Bayesian method, we constructed two artificial data sets. Both data sets consist of a known function corrupted by additive zero mean Gaussian noise. The first data set, shown in Figure (1a) consists of 100 points from a piecewise linear function in which the leftmost portion is corrupted with noise of variance 3 times greater than the rightmost portion. The second data set, shown in Figure (1b) consists of 100 points from the function $g(t) = 4.26(e^{-t} - 4e^{-2t} + 3e^{-3t})$, corrupted by Gaussian noise of constant variance 0.44. We trained a number of models on these data sets, and they provide a typical set of results for the maximum likelihood and Bayesian methods, together with the error bars on the Bayesian solutions. The model architecture used was a 6 deep binary hierarchy of linear experts. In both cases, the ML solutions tend to overfit the noise in the data set. The Bayesian solutions, on the other hand, are both smooth functions which are better approximations to the underlying functions.

### Time Series Prediction

The Bayesian method was also evaluated on a time series prediction problem. This consists of yearly readings of sunspot activity from 1700 to 1979, and was first

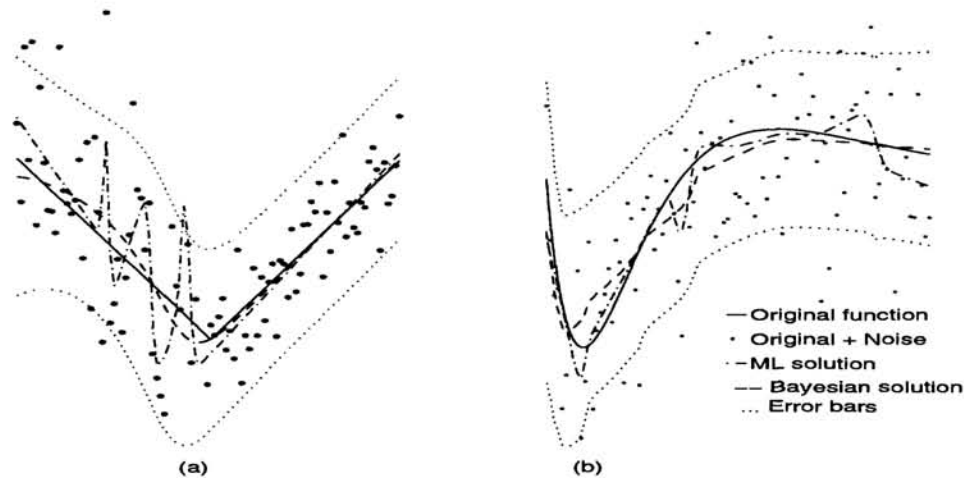

Figure 1: The effect of regularisation on fitting known functions corrupted with noise.

considered in the connectionist community by Weigend, Huberman & Rumelhart (1990), who used an MLP with 8 hidden tanh units, to predict the coming year's activity based on the activities of the previous 12 years. This data set was chosen since it consists of a relatively small number of examples and thus the probability of over-fitting sizeable models is large. In previous work, we considered the use of a mixture of 7 experts on this problem. Due to the problems of over-fitting inherent in ML however, we were constrained to using cross validation to stop the training early. This also constrained the selection of the model order, since the branches of deep networks tend to become "pinched off" during ML training, resulting in local minima during training. The Bayesian method avoids this over-fitting of the gates and allows us to use very large models.

Table 1: Single step prediction on the Sunspots data set using a lag vector of 12 years. NMSE is the mean squared prediction error normalised by the variance of the entire record from 1700 to 1979. The models used were; WHR: Weigend et al's MLP result; 1HME_7_CV: mixture of 7 experts trained via maximum likelihood and using a 10 % cross validation scheme; 8HME2_ML & 8HME2_Bayes: 8 deep binary HME,trained via maximum likelihood (ML) and Bayesian method (Bayes).

| MODEL | Train NMSE | Test NMSE | |
|---|---|---|---|
| | 1700-1920 | 1921-1955 | 1956-1979 |
| WHR | 0.082 | 0.086 | 0.35 |
| 1HME7_CV | 0.061 | 0.089 | 0.27 |
| 8HME2_ML | 0.052 | 0.162 | 0.41 |
| 8HME2_Bayes | 0.079 | 0.089 | 0.26 |

Table 1 shows the results obtained using a variety of methods on the sunspots task. The Bayesian method performs significantly better on the test sets than the maximum likelihood method (8HME2_ML), and is competitive with the MLP of Weigend et al (WHR). It should be noted that even though the number of parameters in the 8 deep binary HME (4992) used is much larger than the number of training examples (209), the Bayesian method still avoids over-fitting of the data. This allows us to specify large models and avoids the need for prior architecture selection, although in some cases such selection may be advantageous, for example if the number of processes inherent in the data is known *a-priori*.

In our experience with linear experts, the smoothness prior on the output function of the expert does not have an important effect; the prior on the gates and the Bayesian inference of the noise level are the important factors. We expect that the smoothness prior would become more important if the experts used more complex basis functions.

## DISCUSSION

The EM algorithm is a special case of the ensemble learning algorithm presented here: the EM algorithm is obtained if we constrain $Q_\Theta(\Theta)$ and $Q_\beta(\beta)$ to be delta functions and fix $\alpha = 0$. The Bayesian ensemble works better because it includes regularization and because the uncertainty of the parameters is taken into account when predictions are made. It could be of interest in future work to investigate how other models trained by EM could benefit from the ensemble learning approach such as hidden Markov models.

The Bayesian method of avoiding over-fitting has been shown to lend itself naturally to the mixture of experts architecture. The Bayesian approach can be implemented practically with only a small computational overhead and gives significantly better performance than the ML model.

## Footnotes

[1] Although $\beta_j$ is a parameter of expert $j$, in common with MacKay (1992*a*) we consider it as a hyperparameter on the Gaussian noise prior.

[2] In all notation, we assume that the input vector is augmented by a constant term, which avoids the need to specify a "bias" term in the parameter vectors.

## References

Dempster, A. P., Laird, N. M. & Rubin, D. B. (1977), 'Maximum likelihood from incomplete data via the EM algorithm', *Journal of the Royal Statistical Society, Series B* **39**, 1–38.

Geman, S., Bienenstock, E. & Doursat, R. (1992), 'Neural networks and the bias / variance dilemma', *Neural Computation* **5**, 1–58.

Gull, S. F. (1989), Developments in maximum entropy data analysis, *in* J. Skilling, ed., 'Maximum Entropy and Bayesian Methods, Cambridge 1988', Kluwer, Dordrecht, pp. 53–71.

Hinton, G. E. & van Camp, D. (1993), Keeping neural networks simple by minimizing the description length of the weights, To appear in: *Proceedings of COLT-93*.

Jordan, M. I. & Jacobs, R. A. (1994), 'Hierarchical Mixtures of Experts and the EM algorithm', *Neural Computation* **6**, 181–214.

MacKay, D. J. C. (1992*a*), 'Bayesian interpolation', *Neural Computation* **4**(3), 415–447.

MacKay, D. J. C. (1992*b*), 'The evidence framework applied to classification networks', *Neural Computation* **4**(5), 698–714.

Neal, R. M. & Hinton, G. E. (1993), 'A new view of the EM algorithm that justifies incremental and other variants'. Submitted to Biometrika. Available at URL:ftp://ftp.cs.toronto.edu/pub/radford/www.

Weigend, A. S., Huberman, B. A. & Rumelhart, D. E. (1990), 'Predicting the future: a connectionist approach', *International Journal of Neural Systems* **1**, 193–209.
